# Large Scale Online Learning.

**Léon Bottou**
NEC Labs America
Princeton NJ 08540
leon@bottou.org

**Yann Le Cun**
NEC Labs America
Princeton NJ 08540
yann@lecun.com

## Abstract

We consider situations where training data is abundant and computing resources are comparatively scarce. We argue that suitably designed online learning algorithms asymptotically outperform any batch learning algorithm. Both theoretical and experimental evidences are presented.

## 1  Introduction

The last decade brought us tremendous improvements in the performance and price of mass storage devices and network systems. Storing and shipping audio or video data is now inexpensive. Network traffic itself provides new and abundant sources of data in the form of server log files. The availability of such large data sources provides clear opportunities for the machine learning community.

These technological improvements have outpaced the exponential evolution of the computing power of integrated circuits (Moore's law). This remark suggests that learning algorithms must process increasing amounts of data using comparatively smaller computing resources.

This work assumes that datasets have grown to practically infinite sizes and discusses which learning algorithms asymptotically provide the best generalization performance using limited computing resources.

- Online algorithms operate by repetitively drawing a fresh random example and adjusting the parameters on the basis of this single example only. Online algorithms can quickly process a large number of examples. On the other hand, they usually are not able to fully optimize the cost function defined on these examples.

- Batch algorithms avoid this issue by completely optimizing the cost function defined on a set of training examples. On the other hand, such algorithms cannot process as many examples because they must iterate several times over the training set to achieve the optimum.

As datasets grow to practically infinite sizes, we argue that online algorithms outperform learning algorithms that operate by repetitively sweeping over a training set.

## 2  Gradient Based Learning

Many learning algorithms optimize an empirical cost function $C_n(\theta)$ that can be expressed as the average of a large number of terms $L(z, \theta)$. Each term measures the cost associated with running a model with parameter vector $\theta$ on independent examples $z_i$ (typically input/output pairs $z_i = (x_i, y_i)$.)

$$C_n(\theta) \triangleq \frac{1}{n} \sum_{i=1}^{n} L(z_i, \theta) \tag{1}$$

Two kinds of optimization procedures are often mentioned in connection with this problem:

- *Batch* gradient: Parameter updates are performed on the basis of the gradient and Hessian information accumulated over a predefined training set:

$$\begin{aligned} \theta(k) &= \theta(k-1) - \Phi_k \frac{\partial C_n}{\partial \theta}(\theta(k-1)) \\ &= \theta(k-1) - \frac{1}{n} \Phi_k \sum_{i=1}^{n} \frac{\partial L}{\partial \theta}(z_i, \theta(k-1)) \end{aligned} \tag{2}$$

  where $\Phi_k$ is an appropriately chosen positive definite symmetric matrix.

- *Online* gradient: Parameter updates are performed on the basis of a single sample $z_t$ picked randomly at each iteration:

$$\theta(t) = \theta(t-1) - \frac{1}{t} \Phi_t \frac{\partial L}{\partial \theta}(z_t, \theta(t-1)) \tag{3}$$

  where $\Phi_t$ is again an appropriately chosen positive definite symmetric matrix. Very often the examples $z_t$ are chosen by cycling over a randomly permuted training set. Each cycle is called an *epoch*. This paper however considers situations where the supply of training samples is practically unlimited. Each iteration of the online algorithm utilizes a fresh sample, unlikely to have been presented to the system before.

Simple batch algorithms converge *linearly*[1] to the optimum $\theta_n^*$ of the empirical cost. Careful choices of $\Phi_k$ make the convergence *super-linear* or even *quadratic*[2] in favorable cases (Dennis and Schnabel, 1983).

Whereas online algorithms may converge to the general area of the optimum at least as fast as batch algorithms (Le Cun et al., 1998), the optimization proceeds rather slowly during the final convergence phase (Bottou and Murata, 2002). The noisy gradient estimate causes the parameter vector to fluctuate around the optimum in a bowl whose size decreases like $1/t$ at best.

Online algorithms therefore seem hopelessly slow. However, the above discussion compares the speed of convergence toward the minimum of the *empirical cost $C_n$*, whereas one should be much more interested in the convergence toward the minimum $\theta^*$ of the *expected cost $C_\infty$*, which measures the generalization performance:

$$C_\infty(\theta) \triangleq \int L(z, \theta)\, p(z)\, \mathrm{d}z \tag{4}$$

Density $p(z)$ represents the unknown distribution from which the examples are drawn (Vapnik, 1974). This is the fundamental difference between *optimization speed* and *learning speed*.

## 3   Learning Speed

Running an efficient batch algorithm on a training set of size $n$ quickly yields the empirical optimum $\theta_n^*$. The sequence of empirical optima $\theta_n^*$ usually converges to the solution $\theta^*$ when the training set size $n$ increases.

In contrast, online algorithms randomly draw one example $z_t$ at each iteration. When these examples are drawn from a set of $n$ examples, the online algorithm minimizes the empirical error $C_n$. When these examples are drawn from the asymptotic distribution $p(z)$, it minimizes the expected cost $C_\infty$. Because the supply of training samples is practically unlimited, each iteration of the online algorithm utilizes a fresh example. These fresh examples follow the asymptotic distribution. The parameter vectors $\theta(t)$ thus directly converge to the optimum $\theta^*$ of the expected cost $C_\infty$.

The convergence speed of the batch $\theta_n^*$ and online $\theta(t)$ sequences were first compared by Murata and Amari (1999). This section reports a similar result whose derivation uncovers a deeper relationship between these two sequences. This approach also provides a mathematically rigorous treatment (Bottou and Le Cun, 2003).

Let us first define the *Hessian* matrix $\mathcal{H}$ and *Fisher information* matrix $\mathcal{G}$:

$$\mathcal{H} \triangleq \mathbf{E}\left(\frac{\partial^2}{\partial\theta\partial\theta}L(\mathbf{z},\theta^*)\right) \qquad \mathcal{G} \triangleq \mathbf{E}\left(\left[\frac{\partial L}{\partial\theta}(\mathbf{z},\theta^*)\right]\left[\frac{\partial L}{\partial\theta}(\mathbf{z},\theta^*)\right]^{\mathsf{T}}\right)$$

Manipulating a Taylor expansion of the gradient of $C_n(\theta)$ in the vicinity of $\theta_{n-1}^*$ immediately provides the following recursive relation between $\theta_n^*$ and $\theta_{n-1}^*$.

$$\theta_n^* = \theta_{n-1}^* - \frac{1}{n}\Psi_n\frac{\partial L}{\partial\theta}(z_n,\theta_{n-1}^*) + \mathcal{O}\left(\frac{1}{n^2}\right) \tag{5}$$

with

$$\Psi_n \triangleq \left(\frac{1}{n}\sum_{i=1}^{n}\frac{\partial^2}{\partial\theta\partial\theta}L(z_i,\theta_{n-1}^*)\right)^{-1} \xrightarrow[t\to\infty]{} \mathcal{H}^{-1}$$

Relation (5) describes the $\theta_n^*$ sequence as a recursive stochastic process that is essentially similar to the online learning algorithm (3). Each iteration of this "algorithm" consists in picking a fresh example $z_n$ and updating the parameters according to (5). This is not a practical algorithm because we have no analytical expression for the second order term. We can however apply the mathematics of online learning algorithms to this stochastic process.

The similarity between (5) and (3) suggests that both the batch and online sequences converge at the same speed for adequate choices of the scaling matrix $\Phi_t$. Under customary regularity conditions, the following asymptotic speed results holds when the scaling matrix $\Phi_t$ converges to the inverse $\mathcal{H}^{-1}$ of the Hessian matrix.

$$\mathbf{E}\left(|\theta(t)-\theta^*|^2\right) + \mathrm{o}\left(\frac{1}{t}\right) = \mathbf{E}\left(|\theta_t^*-\theta^*|^2\right) + \mathrm{o}\left(\frac{1}{t}\right) = \frac{\mathbf{tr}\left(\mathcal{H}^{-1}\,\mathcal{G}\,\mathcal{H}^{-1}\right)}{t} \tag{6}$$

This convergence speed expression has been discovered many times. Tsypkin (1973) establishes (6) for linear systems. Murata and Amari (1999) address generic stochastic gradient algorithms with a constant scaling matrix. Our result (Bottou and Le Cun, 2003) holds when the scaling matrix $\Phi_t$ depends on the previously seen examples, and also holds when the stochastic update is perturbed by unspecified second order terms, as in equation (5). See the appendix for a proof sketch (Bottou and LeCun, 2003).

Result (6) applies to both the online $\theta(t)$ and batch $\theta(t)$ sequences. Not only does it establish that both sequences have $\mathcal{O}\left(1/t\right)$ convergence, but also it provides the value of

the constant. This constant is neither affected by the second order terms of (5) nor by the convergence speed of the scaling matrix $\Phi_t$ toward $\mathcal{H}^{-1}$.

In the Maximum Likelihood case, it is well known that both $\mathcal{H}$ and $\mathcal{G}$ are equal on the optimum. Equation (6) then indicates that the convergence speed saturates the Cramer-Rao bound. This fact was known in the case of the natural gradient algorithm (Amari, 1998). It remains true for a large class of online learning algorithms.

Result (6) suggests that the scaling matrix $\Phi_t$ should be a full rank approximation of the Hessian $\mathcal{H}$. Maintaining such an approximation becomes expensive when the dimension of the parameter vector increases. The computational cost of each iteration can be drastically reduced by maintaining only a coarse approximations of the Hessian (e.g. diagonal, block-diagonal, multiplicative, etc.). A proper setup ensures that the convergence speed remains $\mathcal{O}\left(1/t\right)$ despite a less favorable constant factor.

The similar nature of the convergence of the batch and online sequences can be summarized as follows. Consider two optimally designed batch and online learning algorithms. The best generalization error is asymptotically achieved by *the learning algorithm that uses the most examples* within the allowed time.

## 4   Computational Cost

The discussion so far has established that a properly designed online learning algorithm performs as well as any batch learning algorithm for a same number of examples. We now establish that, given the same computing resources, an online learning algorithm can asymptotically process more examples than a batch learning algorithm.

Each iteration of a batch learning algorithm running on $N$ training examples requires a time $K_1 N + K_2$. Constants $K_1$ and $K_2$ respectively represent the time required to process each example, and the time required to update the parameters. Result (6) provides the following asymptotic equivalence:

$$(\theta_N^* - \theta^*)^2 \ \sim \ \frac{1}{N}$$

The batch algorithm must perform enough iterations to approximate $\theta_N^*$ with at least the same accuracy ($\sim 1/N$). An efficient algorithm with quadratic convergence achieves this after a number of iterations asymptotically proportional to $\log\log N$.

Running an online learning algorithm requires a constant time $K_3$ per processed example. Let us call $T$ the number of examples processed by the online learning algorithm using the same computing resources as the batch algorithm. We then have:

$$K_3 T \sim (K_1 N + K_2)\log\log N \quad \Longrightarrow \quad T \sim N \ \log\log N$$

The parameter $\theta(T)$ of the online algorithm also converges according to (6). Comparing the accuracies of both algorithms shows that the online algorithm asymptotically provides a better solution by a factor $\mathcal{O}\left(\log\log N\right)$.

$$(\theta(T) - \theta^*)^2 \ \sim \ \frac{1}{N \ \log\log N} \ \ll \ \frac{1}{N} \ \sim \ (\theta_N^* - \theta^*)^2$$

This $\log\log N$ factor corresponds to the number of iterations required by the batch algorithm. This number increases slowly with the desired accuracy of the solution. In practice, this factor is much less significant than the actual value of the constants $K_1$, $K_2$ and $K_3$. Experience shows however that online algorithms are considerably easier to implement. Each iteration of the batch algorithm involves a large summation over all the available examples. Memory must be allocated to hold these examples. On the other hand, each iteration of the online algorithm only involves one random example which can then be discarded.

## 5   Experiments

A simple validation experiment was carried out using synthetic data. The examples are input/output pairs $(x, y)$ with $x \in \mathcal{R}^{20}$ and $y = \pm 1$. The model is a single sigmoid unit trained using the least square criterion.

$$L(x, y, \theta) = (1.5y - f(\theta x))^2$$

where $f(x) = 1.71 \tanh(0.66x)$ is the standard sigmoid discussed in LeCun et al. (1998). The sigmoid generates various curvature conditions in the parameter space, including negative curvature and plateaus. This simple model represents well the final convergence phase of the learning process. Yet it is also very similar to the widely used generalized linear models (GLIM) (Chambers and Hastie, 1992).

The first component of the input $x$ is always 1 in order to compensate the absence of a bias parameter in the model. The remaining 19 components are drawn from two Gaussian distributions, centered on $(-1, -1, \ldots, -1)$ for the first class and $(+1, +1, \ldots, +1)$ for the second class. The eigenvalues of the covariance matrix of each class range from 1 to 20.

Two separate sets for training and testing were drawn with 1 000 000 examples each. One hundred permutations of the first set are generated. Each learning algorithm is trained using various number of examples taken sequentially from the beginning of the permuted sets. The resulting performance is then measured on the testing set and averaged over the one hundred permutations.

**Batch-Newton algorithm**

The reference batch algorithm uses the Newton-Raphson algorithm with Gauss-Newton approximation (Le Cun et al., 1998). Each iteration visits all the training and computes both gradient $g$ and the Gauss-Newton approximation $H$ of the Hessian matrix.

$$g = \sum_i \frac{\partial L}{\partial \theta}(x_i, y_i, \theta_{k-1}) \qquad H = \sum_i \left(f'(\theta_{k-1}x_i)\right)^2 x_i x_i^{\mathsf{T}}$$

The parameters are then updated using Newton's formula:

$$\theta_k = \theta_{k-1} - H^{-1}g$$

Iterations are repeated until the parameter vector moves by less than $0.01/N$ where $N$ is the number of training examples. This algorithm yields quadratic convergence speed.

**Online-Kalman algorithm**

The online algorithm performs *a single sequential sweep* over the training examples. The parameter vector is updated after processing each example $(x_t, y_t)$ as follows:

$$\theta_t = \theta_{t-1} - \frac{1}{\tau} \Phi_t \frac{\partial L}{\partial \theta}(x_t, y_t, \theta_{t-1})$$

The scalar $\tau = \max(20, t - 40)$ makes sure that the first few examples do not cause impractically large parameter updates. The scaling matrix $\Phi_t$ is equal to the inverse of a leaky average of the per-example Gauss-Newton approximation of the Hessian.

$$\Phi_t = \left( \left(1 - \frac{2}{\tau}\right) \Phi_{t-1}^{-1} + \left(\frac{2}{\tau}\right) \left(f'(\theta_{t-1}x_t)\right)^2 x_t x_t^{\mathsf{T}} \right)^{-1}$$

The implementation avoids the matrix inversions by directly computing $\Phi_t$ from $\Phi_{t-1}$ using the matrix inversion lemma. (see (Bottou, 1998) for instance.)

$$\left(\alpha A^{-1} + \beta uu^{\mathsf{T}}\right)^{-1} = \frac{1}{\alpha}\left(A - \frac{(Au)(Au)^{\mathsf{T}}}{\alpha/\beta + u^{\mathsf{T}}Au}\right)$$

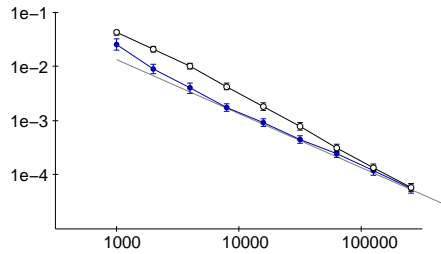
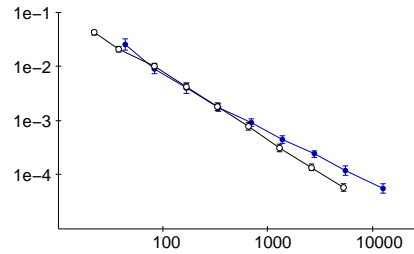

Figure 1: Average $(\theta - \theta^*)^2$ as a function of the number of examples. The gray line represents the theoretical prediction (6). Filled circles: batch. Hollow circles: online. The error bars indicate a 95% confidence interval.

Figure 2: Average $(\theta - \theta^*)^2$ as a function of the training time (milliseconds). Hollow circles: online. Filled circles: batch. The error bars indicate a 95% confidence interval.

The resulting algorithm slightly differs from the Adaptive Natural Gradient algorithm (Amari, Park, and Fukumizu, 1998). In particular, there is little need to adjust a learning rate parameter in the Gauss-Newton approach. The $1/t$ (or $1/\tau$) schedule is asymptotically optimal.

**Results**

The optimal parameter vector $\theta^*$ was first computed on the testing set using the batch-newton approach. The matrices $\mathcal{H}$ and $\mathcal{G}$ were computed on the testing set as well in order to determine the constant in relation (6).

Figure 1 plots the average squared distance between the optimal parameter vector $\theta^*$ and the parameter vector $\theta$ achieved on training sets of various sizes. The gray line represents the theoretical prediction. Both the batch points and the online points join the theoretical prediction when the training set size increases. Figure 2 shows the same data points as a function of the CPU time required to run the algorithm on a standard PC. The online algorithm gradually becomes more efficient when the training set size increases. This happens because the batch algorithm needs to perform additional iterations in order to maintain the same level of accuracy.

In practice, the test set mean squared error (MSE) is usually more relevant than the accuracy of the parameter vector. Figure 3 displays a logarithmic plot of the difference between the MSE and the best achievable MSE, that is to say the MSE achieved by parameter vector $\theta^*$. This difference can be approximated as $(\theta - \theta^*)^\mathsf{T}\mathcal{H}(\theta - \theta^*)$. Both algorithms yield virtually identical errors for the same training set size. This suggests that the small differences shown in figure 1 occur along the low curvature directions of the cost function. Figure 4 shows the MSE as a function of the CPU time. The online algorithm always provides higher accuracy in significantly less time.

As expected from the theoretical argument, the online algorithm asymptotically outperforms the super-linear Newton-Raphson algorithm[3]. More importantly, the online algorithm achieves this result by performing *a single sweep* over the training data. This is a very significant advantage when the data does not fit in central memory and must be sequentially accessed from a disk based database.

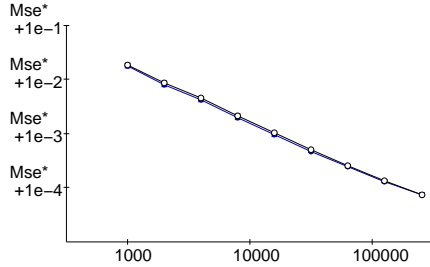

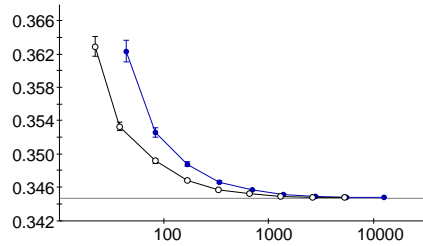

Figure 3: Average test MSE as a function of the number of examples (left). The vertical axis shows the logarithm of the difference between the error and the best error achievable on the testing set. Both curves are essentially superposed.

Figure 4: Average test MSE as a function of the training time (milliseconds). Hollow circles: online. Filled circles: batch. The gray line indicates the best mean squared error achievable on the test set.

## 6    Conclusion

Many popular algorithms do not scale well to large number of examples because they were designed with small data sets in mind. For instance, the training time for Support Vector Machines scales somewhere between $N^2$ and $N^3$, where $N$ is the number of examples. Our baseline super-linear batch algorithm learns in $N \log \log N$ time. We demonstrate that adequate online algorithms asymptotically achieve the same generalization performance in $N$ time after a single sweep on the training set.

The convergence of learning algorithms is usually described in terms of a *search* phase followed by a *final convergence* phase (Bottou and Murata, 2002). Solid empirical evidence (Le Cun et al., 1998) suggests that online algorithms outperform batch algorithms during the search phase. The present work provides both theoretical and experimental evidence that an adequate online algorithm outperforms any batch algorithm during the final convergence phase as well.

## Appendix[4]: Sketch of the convergence speed proof

**Lemma** — Let $(u_t)$ be a sequence of positive reals verifying the following recurrence:

$$u_t = \left(1 - \frac{\alpha}{t} + o\left(\frac{1}{t}\right)\right) u_{t-1} + \frac{\beta}{t^2} + o\left(\frac{1}{t^2}\right) \qquad (7)$$

The lemma states that $t\, u_t \longrightarrow \frac{\beta}{\alpha-1}$ when $\alpha > 1$ and $\beta > 0$. The proof is delicate because the result holds regardless of the unspecified low order terms of the recurrence. However, it is easy to illustrate this convergence with simple numerical simulations.

**Convergence speed** — Consider the following recursive stochastic process:

$$\theta(t) \;=\; \theta(t-1) - \frac{1}{t}\, \Phi_t \frac{\partial L}{\partial \theta}(z_t, \theta(t-1)) + \mathcal{O}\left(\frac{1}{n^2}\right) \qquad (8)$$

Our discussion addresses the final convergence phase of this process. Therefore we assume that the parameters $\theta$ remain confined in a bounded domain $\mathcal{D}$ where the cost function $C_\infty(\theta)$ is convex and has a single non degenerate minimum $\theta^* \in \mathcal{D}$. We can assume

$\theta^* = 0$ without loss of generality. We write $\mathbf{E}_t\,(X)$ the conditional expectation of $X$ given all that is known before time $t$, including the initial conditions $\theta_0$ and the selected examples $\mathbf{z}_1, \ldots, \mathbf{z}_{t-1}$. We initially assume also that $\Phi_t$ is a function of $\mathbf{z}_1, \ldots, \mathbf{z}_{t-1}$ only.

Using (8), we write $\mathbf{E}_t\,(\theta_t \theta_t')$ as a function of $\theta_{t-1}$. Then we simplify[5] and take the trace.

$$\mathbf{E}_t\left(\,|\theta_t|^2\right) \;=\; |\theta_{t-1}|^2 \;-\; \frac{2}{t}\,|\theta_{t-1}|^2 \;+\; \mathrm{o}\left(\frac{|\theta_{t-1}|^2}{t}\right) \;+\; \frac{\mathbf{tr}\left(\mathcal{H}^{-1}\,\mathcal{G}\,\mathcal{H}^{-1}\right)}{t^2} \;+\; \mathrm{o}\left(\frac{1}{t^2}\right)$$

Taking the unconditional expectation yields a recurence similar to (7). We then apply the lemma and conclude that $t\,\mathbf{E}(|\theta_t|^2) \longrightarrow \mathbf{tr}\left(\mathcal{H}^{-1}\,\mathcal{G}\,\mathcal{H}^{-1}\right)$.

**Remark 1** — The notation $\mathrm{o}\,(X_t)$ is quite ambiguous when dealing with stochastic processes. There are many possible flavors of convergence, including uniform convergence, almost sure convergence, convergence in probability, etc. Furthermore, it is not true in general that $\mathbf{E}\,(\mathrm{o}\,(X_t)) = \mathrm{o}\,(\mathbf{E}\,(X_t))$. The complete proof precisely defines the meaning of these notations and carefully checks their properties.

**Remark 2** — The proof sketch assumes that $\Phi_t$ is a function of $\mathbf{z}_1, \ldots, \mathbf{z}_{t-1}$ only. In (5), $\Psi_t$ also depends on $\mathbf{z}_t$. The result still holds because the contribution of $\mathbf{z}_t$ vanishes quickly when $t$ grows large.

**Remark 3** — The same $\frac{1}{t}$ behavior holds when $\Phi_t \to \Phi^*$ and when $\Phi^*$ is greater than $\frac{1}{2}\mathcal{H}^{-1}$ in the semi definite sense. The constant however is worse by a factor roughly equal to $||H\Phi^*||$.

### Acknowledgments

The authors acknowledge extensive discussions with Yoshua Bengio, Sami Bengio, Ronan Collobert, Noboru Murata, Kenji Fukumizu, Susanna Still, and Barak Pearlmutter.

## Footnotes

[1] Linear convergence speed: $\log 1/|\theta(k) - \theta_n^*|^2$ grows linearly with $k$.

[2] Quadratic convergence speed: $\log\log 1/|\theta(k) - \theta_n^*|^2$ grows linearly with $k$.

[3]Generalized linear models are usually trained using the IRLS method (Chambers and Hastie, 1992) which is closely related to the Newton-Raphson algorithm and requires similar computational resources.

[4]This section has been added for the final version

[5]Recall $\mathbf{E}_t\left(\Phi_t \frac{\partial L}{\partial \theta}(\mathbf{z}_t, \theta)\right) = \Phi_t \frac{\partial C}{\partial \theta}(\theta) = \Phi_t \mathcal{H}\theta + \mathrm{o}\,(|\theta|) = \theta + \mathrm{o}\,(|\theta|)$

### References

Amari, S. (1998). Natural Gradient Works Effi ciently in Learning. *Neural Computation*, 10(2):251–276.

Bottou, L. (1998). Online Algorithms and Stochastic Approximations, 9-42. In Saad, D., editor, *Online Learning and Neural Networks*. Cambridge University Press, Cambridge, UK.

Bottou, L. and Murata, N. (2002). Stochastic Approximations and Effi cient Learning. In Arbib, M. A., editor, *The Handbook of Brain Theory and Neural Networks, Second edition,*. The MIT Press, Cambridge, MA.

Bottou, L. and Le Cun, Y. (2003). Online Learning for Very Large Datasets. NEC Labs TR-2003-L039. To appear: *Applied Stochastic Models in Business and Industry*. Wiley.

Chambers, J. M. and Hastie, T. J. (1992). *Statistical Models in S,* Chapman & Hall, London.

Dennis, J. and Schnabel, R. B. (1983). *Numerical Methods For Unconstrained Optimization and Nonlinear Equations*. Prentice-Hall, Inc., Englewood Cliffs, New Jersey.

Amari, S. and Park, H. and Fukumizu, K. (1998). Adaptive Method of Realizing Natural Gradient Learning for Multilayer Perceptrons, *Neural Computation*, 12(6):1399–1409

Le Cun, Y., Bottou, L., Orr, G. B., and Müller, K.-R. (1998). Effi cient Back-prop. In *Neural Networks, Tricks of the Trade*, Lecture Notes in Computer Science 1524. Springer Verlag.

Murata, N. and Amari, S. (1999). Statistical analysis of learning dynamics. *Signal Processing*, 74(1):3–28.

Vapnik, V. N. and Chervonenkis, A. (1974). *Theory of Pattern Recognition* (in russian). Nauka.

Tsypkin, Ya. (1973). *Foundations of the theory of learning systems*. Academic Press.

